# Dopaminergic Neuromodulation Brings a Dynamical Plasticity to the Retina

Eric Boussard            Jean-François Vibert

B3E, INSERM U263
Faculté de médecine Saint-Antoine
27 rue Chaligny
75571 Paris cedex 12

## Abstract

The fovea of a mammal retina was simulated with its detailed biological properties to study the local preprocessing of images. The direct visual pathway (photoreceptors, bipolar and ganglion cells) and the horizontal units, as well as the D-amacrine cells were simulated. The computer program simulated the analog non-spiking transmission between photoreceptor and bipolar cells, and between bipolar and ganglion cells, as well as the gap-junctions between horizontal cells, and the release of dopamine by D-amacrine cells and its diffusion in the extra-cellular space. A 64×64 photoreceptors retina, containing 16,448 units, was carried out. This retina displayed contour extraction with a Mach effect, and adaptation to brightness. The simulation showed that the dopaminergic amacrine cells were necessary to ensure adaptation to local brightness.

## 1    INTRODUCTION

The retina is the first stage in visual information processing. One of its functions is to compress the information received from the environment by removing spatial and temporal redundancies that occur in the light input signal. Modelling and computer simulations present an efficient means to investigate and characterize the physiological mechanisms that underlie such a complex process. In fact, filtering depends on the quality of the input image (van Hateren, 1992):

1.**High mean light intensity** (high signal to noise ratio). A high-pass filter enhances the edges (contour extraction) and the temporal changes of the input.

2.**Low mean light intensity** (low signal to noise ratio). The sensitivity of high-pass filters to noise makes them inefficient in this case. A low-pass filter, averaging the signal over several receptors, is required to extract the relevant information.

There are three aspects in the filtering adaptivity displayed by the retina: adaptivity to i) the global spatial changes in the image, ii) the local spatial changes in the image, iii) the temporal changes in the image. We will focus on the second feature. A biologically plausible mammalian retina was modelled and simulated to explore the local preprocessing of the images. A first model (Bedfer & Vibert, 1992), that did not take into account the dopamine neuromodulation, reproduced some of the behaviors found in the living retina, like a progressive decrease of ganglion cells' firing rate in response to a constant image presented to photoreceptors, reversed post-image, and optic illusion (Hermann grid). The model, however, displayed a poor local adaptivity. It could not give both a good contrast rendering and a Mach effect. The Mach effect is a psychophysical law that is characterized by an edge enhancement (Ratliff, 1965). The retina network produces a double lighter and darker contour from the frontier line between two areas of different brightness in the stimulus. This phenomenon is indispensable for contour extraction. This paper will first present the conditions in which high-pass filtering and low-pass filtering occur exclusively in the retina model. These results are then compared to those obtained with a model that includes dopamine neuromodulation, thus illustrating the role played by dopamine in local adaptivity (Besharse & Iuvone, 1992).

## 2   METHODS

The retina is an unusual neural structure: i) the photoreceptors respond to light by an hyperpolarization, ii) signal transmission from photoreceptors to bipolar units does not involve spikes, neurotransmitter release at these synapses is a continuous function of the membrane potential (Buser & Imbert, 1987). Only ganglion cells generate spikes. Furthermore, horizontal cells are connected by dopamine dependent gap-junctions. Dopamine is an ubiquitous neurotransmitter and neuromodulator in the central nervous system. In the visual pathway, dopamine affects several types of retinal neurons (Witkovsky & Dearry, 1992). Dopamine is released by stimulated D-amacrine and interplexiform cells. It diffuses in the extra-cellular space, and produces: cone shortening and rod elongation, reduced permeability of gap-junctions, increased conductance of glutamate-induced current among horizontal cells, increased conductance of the cone-to- horizontal cell synapse, and retro-inhibition on D-amacrine cells (Djamgoz & Wagner, 1992). Our model focused on the adaptive filtering mechanism in the fovea that enables the retina to simultaneously perform both high-pass and low-pass filtering. Therefore, dopamine action on gap-junction between horizontal cells and the retro-inhibition on D-amacrine cells was the only dopamine effect implemented (fig. 1). Our model included the three neuron types of the direct pathway – photoreceptors, bipolar and ganglion units – as well as two types of the indirect pathway – the horizontal and dopaminergic amacrine cells. Only the On pathway of a mammal fovea was studied here. Each neuron type has been modelled with its own anatomical and electrophysiolog-

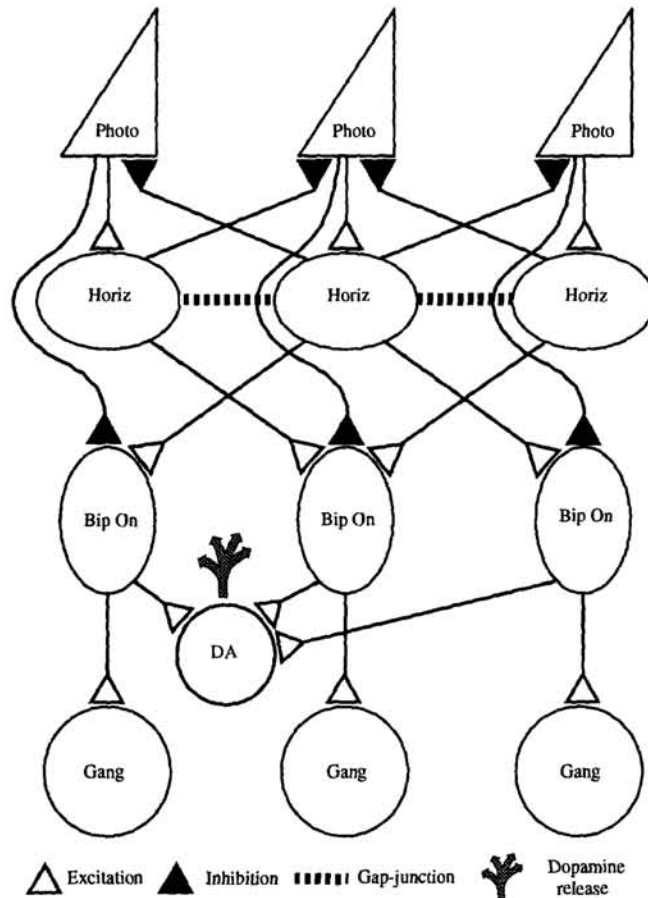

Figure 1: The dopaminergic amacrine units in the modelled retina.

*The connections of an On center pathway in the simulated retina. Photo: Photoreceptors. Horiz: Horizontal units. Bip: Bipolar units. Gang: Ganglion Units. DA: Dopaminergic Amacrine unit. DA units are stimulated by many bipolar units. With an enough excitation, they can release dopamine in the extracellular space. This released dopamine goes to modulates the conductance value of horizontal gap-junctions.*

ical properties (Wässle & Boycott, 1991)(Lewick & Dvorak, 1986). The temporal evolution of the membrane potential of each unit can be recorded.

# 3    RESULTS

A 64x64 photoreceptors retina was constructed as a noisy hexagonal frame where photoreceptors, bipolar and ganglion units were connected to their nearest neighbours. Horizontal units were connected to their 18 nearest photoreceptors and bipolar units, with a number of synaptic boutons decreasing as a function of distance. They did not retroact on the nearest photoreceptor. This horizontal layer architecture produces lateral inhibition. Each modelled D-amacrine unit was connected to about fifty bipolar units. The diffusion of released dopamine in the extra-cellular space was simulated. The modelled retina consisted of 16,448 units and 862,720 synapses.

At each simulation, the photoreceptors layer was stimulated by an input image. Stimulations were given as a 256×256 pixel image presented to the simulated 64×64 photoreceptor retina. Since the localization of photoreceptors was not regular, each receptor received the input from 16 pixels on the average. The output image was reconstructed using the ganglion units response. For each of the 4096 ganglion units the spike frequency was measured during a given time (according to the experiment) and coded in a grey level for the given unit retinotopic position. Thus, each simulation produced an image of the retina output. This output image was compared to the input image.

The input image (stimulus) consisted here of one white disk on a dark background. The results presented, in fig. 2, were obtained after 750 ms of stationary stimulations. The stimuli were here a white disk on a black background. The inputs were stationary to avoid temporal effects owing to evolving inputs. Output images of stationary inputs, however, vanished after 1000 ms. The time was limited to 750 ms to optimize the quality of the output image.

Biological datas available on the conductance value suggest that in the mammalian retina the conductance does not remain constant and undergoes a dynamical tuning depending on the local brightness [?]. This provides a range of possible values for the conductance. The behavior of the model was tested for values within this range. Different values lead to different network behaviors. Three types of results were obtained from the simulations :

1. Without dopamine action, the conductance values were fixed for all gap-junctions to $10^{-6}$S (fig. 2-A). The output image rendered well the contrast in the input image, but did not display the Mach effect (low-pass filtering).

2. Without dopamine action, the conductance values were fixed for all gap-junctions to $10^{-10}$S (fig. 2-B). The low conductance value allowed a pronounced Mach effect, but the contrast in the output image was strongly diminished (high-pass filtering). This contrast appears like an average of the two brightness. Only the contour delimited by Mach effect allows the disk to be distinguished.

3. With dopamine, the conductance values were initially set to $10^{-7}$S (fig. 2-C). The output image displayed both the contrast rendering and the Mach effect (locally

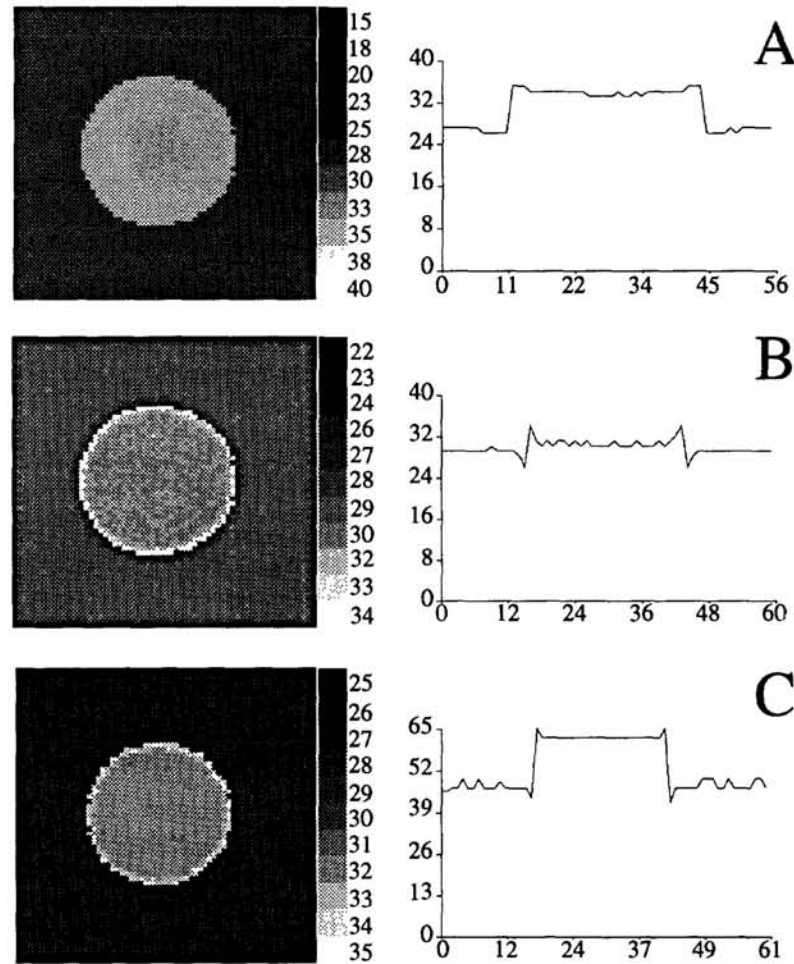

Figure 2: Contour extraction (Mach effect) according to gap-junctions conductances.

*On the left, results obtained after 750 ms of stimulation for an image of a white disk on a black background. On the right, sections through the corresponding image. Abscissa: spike count; Ordinates: geographic position of the unit, from the left side to the middle of the left panel. A: without dopamine (fixed $G_{gap} = 10^{-6}S$). B: without dopamine (fixed $G_{gap} = 10^{-10}S$). C: with dopamine release (starting $G_{gap} = 10^{-7}S$). A gives a good contrast rendering, but no Mach effect. B gives a Mach effect, but there is an averaging between darker and lighter areas. C, with dopaminergic neuromodulation, gives both a Mach effect and a good contrast rendering.*

adaptive filtering).

## 4 DISCUSSION

These results show that the conductance cannot be fixed at a single value for all the gap-junctions. If the conductance value is high (fig. 2-A), the model acts like a low-pass filter. A good contrast rendering was obtained, but there was no Mach effect. If the conductance value is low (fig. 2-B), the model becomes a high-pass filter. A Mach effect was obtained, but the contrast in the post-retinal image was dramatically deteriorated: an undesirable averaging of the brightness between the darker and the more illuminated areas appeared. Therefore in this model the Mach effect was only obtained at the expense of the contrast. A mammalian retina is able to perform both contrast rendering and contour extraction functions together. It works like an adaptive filter. To obtain a similar result, it is necessary to have a variable communication between horizontal units. The simulated retina needs low gap-junctions conductance in the high light intensity areas and high conductance in the low light intensity areas. The conductance of each gap-junction must be tuned according to the local stimulation. The model used to obtain the fig. 2-C takes into account the dopamine release by the D-amacrine cells. Here, the network performs the two antagonist functions of filtering. Dopamine provides our model with the capacity to have a biological behaviour. What is the action of dopamine on network? Dopamine is released by D-amacrine units. Then, it diffuses from its release point into the extra-cellular space among the neurons, reaches gap-junctions and decreases their conductance value. Thus the conductance modulation depends in time and in intensity on the distance between gap-junction and D-amacrine unit. In addition, this action is transient.

## 5 CONCLUSION

Thanks to dopamine neuromodulation, the network is able to subdivise itself into several subnetworks, each having the appropriate gap-junction conductance. Each subnetwork is thus adapted for a better processing of the external stimulus. Dopamine neuromodulation is a chemically addressed system, it acts more diffusely and more slowly than transmission through the axo-synaptic connection system. Therefore neuromodulation adds a dynamical plasticity to the network.

### References

G. Bedfer & J.-F. Vibert. (1992) Image preprocessing in simulated biological retina. *Proc. 14th Ann. Conf. IEEE EMBS* 1570-1571.

J. Besharse & P. Iuvone. (1992). Is dopamine a light-adaptive or a dark-adaptive modulator in retina? *NeuroChemistry International* 20:193-199.

P. Buser & M. Imbert. (1987) *Vision*. Paris: Hermann.

M. Djamgoz & H.-J. Wagner. (1992) Localization and function of dopamine in the adult vertebrate retina. *NeuroChemistry International* 20:139-191.

L. Dowling. (1986) Dopamine: a retinal neuromodulator? *Trends In NeuroSciences*

9:236-240.

W. Levick & D. Dvorak. (1986) The retina – from molecule to network. *Trends In NeuroSciences* **9**:181-185.

F. Ratliff. (1965) *Mach bands: quantitative studies on neural network in the retina.* Holden-Day.

J. H. van Hateren. (1992) Real and optimal images in early vision. *Nature* **360**:68-70.

H. Wässle & B. B. Boycott. (1991) Functional architecture of the mammalian retina. *Physiological Reviews* **71**(2):447-479.

P. Witkovsky & A. Dearry. (1992) Functional roles of dopamine in the vertebrate retina. *Retinal Research* **11**:247-292.